# An entropic estimator for structure discovery

**Matthew Brand**
Mitsubishi Electric Research Laboratories, 201 Broadway, Cambridge MA 02139
brand@merl.com

## Abstract

We introduce a novel framework for simultaneous structure and parameter learning in hidden-variable conditional probability models, based on an entropic prior and a solution for its maximum *a posteriori* (MAP) estimator. The MAP estimate minimizes uncertainty in all respects: cross-entropy between model and data; entropy of the model; entropy of the data's descriptive statistics. Iterative estimation extinguishes weakly supported parameters, compressing and sparsifying the model. Trimming operators accelerate this process by removing excess parameters and, unlike most pruning schemes, guarantee an increase in posterior probability. *Entropic estimation* takes a overcomplete random model and simplifies it, inducing the structure of relations between hidden and observed variables. Applied to hidden Markov models (HMMs), it finds a concise finite-state machine representing the hidden structure of a signal. We entropically model music, handwriting, and video time-series, and show that the resulting models are highly concise, structured, predictive, and *interpretable*: Surviving states tend to be highly correlated with meaningful partitions of the data, while surviving transitions provide a low-perplexity model of the signal dynamics.

## 1 An entropic prior

In entropic estimation we seek to maximize the information content of parameters. For conditional probabilities, parameters values near chance add virtually no information to the model, and are therefore wasted degrees of freedom. In contrast, parameters near the extrema $\{0, 1\}$ are informative because they impose strong constraints on the class of signals accepted by the model. In Bayesian terms, our prior should assert that parameters that do not reduce uncertainty are improbable. We can capture this intuition in a surprisingly simple form: For a model of $N$ conditional probabilities $\boldsymbol{\theta} = \{\theta_1, \ldots, \theta_N\}$ we write

$$P_e(\boldsymbol{\theta}) = \boldsymbol{\theta}^{\boldsymbol{\theta}} = \prod_i^N \theta_i^{\theta_i} = \exp\left[\sum_i^N \theta_i \log \theta_i\right] = e^{-H(\boldsymbol{\theta})} \tag{1}$$

whence we can see that the prior measures a model's freedom from ambiguity ($H(\boldsymbol{\theta})$ is an entropy measure). Applying $P_e(\cdot)$ to a multinomial yields the posterior

$$P_e(\boldsymbol{\theta}|\boldsymbol{\omega}) \propto \frac{P(\boldsymbol{\omega}|\boldsymbol{\theta})P_e(\boldsymbol{\theta})}{P(\boldsymbol{\omega})} \propto \left[\prod_i \theta_i^{\omega_i}\right] \frac{P_e(\boldsymbol{\theta})}{P(\boldsymbol{\omega})} \propto \prod_i \theta_i^{\theta_i + \omega_i} \tag{2}$$

where $\omega_i$ is evidence for event type $i$. With extensive evidence this distribution converges to "fair"(ML) odds for $\boldsymbol{\omega}$, but with scant evidence it skews to stronger odds.

## 1.1  MAP estimator

To obtain MAP estimates we set the derivative of log-posterior to zero, using Lagrange multipliers to ensure $\sum_i \theta_i = 1$,

$$
0 = \frac{\partial}{\partial \theta_i}\left(\log \prod_i \theta_i^{\omega_i + \theta_i} + \lambda\left(\sum_i \theta_i - 1\right)\right) = \sum_i \frac{\partial}{\partial \theta_i}(\omega_i + \theta_i)\log \theta_i + \lambda \sum_i \frac{\partial}{\partial \theta_i}\theta_i
$$

$$
= 1 + \frac{\omega_i}{\theta_i} + \log \theta_i + \lambda \qquad (3)
$$

We obtain $\theta_i$ by working backward from the Lambert $W$ function, a multi-valued inverse function satisfying $W(x)e^{W(x)} = x$. Taking logarithms and setting $y = \log x$,

$$
0 = -W(x) - \log W(x) + \log x = -W(e^y) - \log W(e^y) + y
$$

$$
= \frac{-1}{1/W(e^y)} + \log 1/W(e^y) + \log z + y - \log z
$$

$$
= \frac{-z}{z/W(e^y)} + \log z/W(e^y) + y - \log z \qquad (4)
$$

Setting $\theta_i = z/W(e^y)$, $y = 1 + \lambda + \log z$, and $z = -\omega_i$, eqn. 4 simplifies to eqn. 3, implying

$$
\hat{\theta}_i = \frac{-\omega_i}{W(-\omega_i e^{1+\lambda})} \qquad (5)
$$

Equations 3 and 5 together yield a quickly converging fix-point equation for $\lambda$ and therefore for the entropic MAP estimate. Solutions lie in the $W_{-1}$ branch of Lambert's function. See [Brand, 1997] for methods we developed to calculate the little-known $W$ function.

## 1.2  Interpretation

The negated log-posterior is equivalent to a sum of entropies:

$$
-\log \prod_i \theta_i^{\theta_i + \omega_i} = -\sum_i (\theta_i + \omega_i)\log \theta_i
$$

$$
= -\sum_i (\theta_i \log \theta_i + \omega_i \log \theta_i - \omega_i \log \omega_i + \omega_i \log \omega_i)
$$

$$
= -\sum_i \theta_i \log \theta_i + \sum_i \omega_i \log \frac{\omega_i}{\theta_i} - \sum_i \omega_i \log \omega_i
$$

$$
= H(\boldsymbol{\theta}) + D(\boldsymbol{\omega}\|\boldsymbol{\theta}) + H(\boldsymbol{\omega}) \qquad (6)
$$

Maximizing $P_e(\boldsymbol{\theta}|\boldsymbol{\omega})$ minimizes entropy in all respects: the parameter entropy $H(\boldsymbol{\theta})$; the cross-entropy $D(\boldsymbol{\omega}\|\boldsymbol{\theta})$ between the parameters $\boldsymbol{\theta}$ and the data's descriptive statistics $\boldsymbol{\omega}$; and the entropy of those statistics $H(\boldsymbol{\omega})$, which are calculated relative to the structure of the model. Equivalently, the MAP estimator minimizes the expected coding length, making it a maximally efficient compressor of messages consisting of the model and the data coded relative to the model. Since compression involves separating essential from accidental structure, this can be understood as a form of noise removal. Noise inflates the apparent entropy of a sampled process; this systematically biases maximum likelihood (ML) estimates toward weaker odds, more so in smaller samples. Consequently, the entropic prior is a countervailing bias toward stronger odds.

## 1.3 Model trimming

Because the prior rewards sparse models, it is possible to remove weakly supported parameters from the model while improving its posterior probability, such that $P_e(\boldsymbol{\theta}\backslash\theta_i|\boldsymbol{X}) > P_e(\boldsymbol{\theta}|\boldsymbol{X})$. This stands in contrast to most pruning schemes, which typically try to minimize damage to the posterior. Expanding via Bayes rule and taking logarithms we obtain

$$h_i(\theta_i) = H(\boldsymbol{\theta}) - H(\boldsymbol{\theta}\backslash\theta_i) > \log P(\boldsymbol{X}|\boldsymbol{\theta}) - \log P(\boldsymbol{X}|\boldsymbol{\theta}\backslash\theta_i) \tag{7}$$

where $h_i(\theta_i)$ is the entropy due to $\theta_i$. For small $\theta_i$, we can approximate via differentials:

$$\theta_i \frac{\partial H(\boldsymbol{\theta})}{\partial \theta_i} > \theta_i \frac{\partial \log P(\boldsymbol{X}|\boldsymbol{\theta})}{\partial \theta_i} \tag{8}$$

By mixing the left- and right-hand sides of equations 7 and 8, we can easily identify trimmable parameters—those that contribute more to the entropy than the log-likelihood. E.g., for multinomials we set $h_i(\theta_i) = -\theta_i \log \theta_i$ against r.h.s. eqn. 8 and simplify to obtain

$$\theta_i < \exp\left[-\frac{\partial \log P(\boldsymbol{X}|\boldsymbol{\theta})}{\partial \theta_i}\right] \tag{9}$$

Parameters can be trimmed at any time during training; at convergence trimming can bump the model out of a local probability maximum, allowing further training in a lower-dimensional and possibly smoother parameter subspace.

## 2 Entropic HMM training and trimming

In entropic estimation of HMM transition probabilities, we follow the conventional E-step, calculating the probability mass for each transition to be used as evidence $\boldsymbol{\omega}$:

$$\gamma_{j,i} = \sum_{t}^{T-1} \alpha_j(t) P_{i|j} p_i(x_{t+1}) \beta_i(t+1) \tag{10}$$

where $P_{i|j}$ is the current estimate of the transition probability from state $j$ to state $i$; $p_i(x_{t+1})$ is the output probability of observation $x_{t+1}$ given state $i$, and $\boldsymbol{\alpha}, \boldsymbol{\beta}$ are obtained from forward-backward analysis and follow the notation of Rabiner [1989]. For the M-step, we calculate new estimates $\{\hat{P}_{i|j}\}_i = \boldsymbol{\theta}$ by applying the MAP estimator in §1.1 to each $\boldsymbol{\omega} = \{\gamma_{j,i}\}_i$. That is, $\boldsymbol{\omega}$ is a vector of the evidence for each kind of transition out of a single state; from this evidence the MAP estimator calculates probabilities $\boldsymbol{\theta}$. (In Baum-Welch re-estimation, the maximum-likelihood estimator simply sets $\hat{P}_{i|j} = \gamma_{j,i}/\sum_i \gamma_{j,i}$.)

In iterative estimation, e.g., expectation-maximization (EM), the entropic estimator drives weakly supported parameters toward zero, skeletonizing the model and concentrating evidence on surviving parameters until their estimates converge to near the ML estimate. Trimming appears to accelerate this process by allowing slowly dying parameters to leapfrog to extinction. It also averts numerical underflow errors.

For HMM transition parameters, the trimming criterion of eqn. 9 becomes

$$P_{i|j} < \exp\left[-\frac{\sum_{t=1}^{T-1} \alpha_j(t) p(x_{t+1}|s_i)\beta_i(t+1)}{\sum_k^N \alpha_k(T)}\right] = \exp\left[-\sum_{t=1}^{T-1} \gamma_j(t)\right] \tag{11}$$

where $\gamma_j(t)$ is the probability of state $j$ at time $t$. The multinomial output distributions of a discrete-output HMM can be entropically re-estimated and trimmed in the same manner.

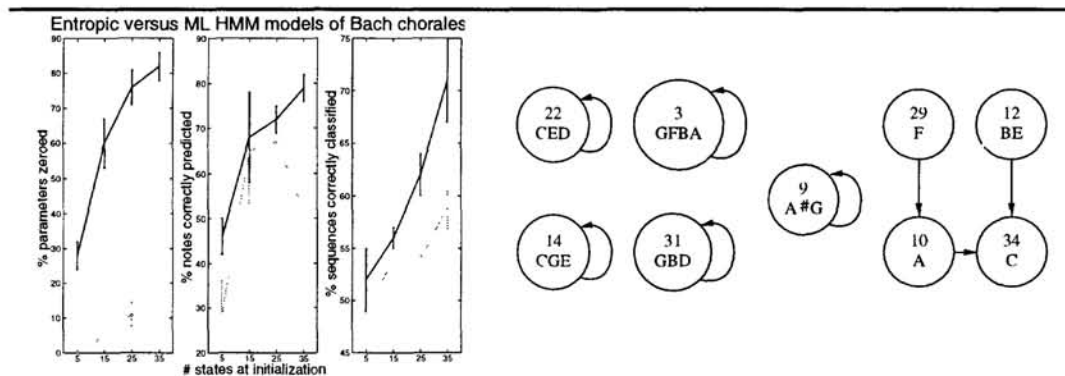

Figure 1: **Left:** Sparsification, classification, and prediction superiority of entropically estimated HMMs modeling Bach chorales. Lines indicate mean performance over 10 trials; error bars are 2 standard deviations. **Right:** High-probability states and subgraphs of interest from an entropically estimated 35-state chorale HMM. Tones output by each state are listed in order of probability. Extraneous arcs have been removed for clarity.

## 3 Structure learning experiments

To explore the practical utility of this framework, we will use entropically estimated HMMs as a window into the hidden structure of some human-generated time-series.

**Bach Chorales:** We obtained a dataset of melodic lines from 100 of J.S. Bach's 371 surviving chorales from the UCI repository [Merz and Murphy, 1998], and transposed all into the key of C. We compared entropically and conventionally estimated HMMs in prediction and classification tasks, training both from identical random initial conditions and trying a variety of different initial state-counts. We trained with 90 chorales and testing with the remaining 10. In ten trials, all chorales were rotated into the test set. Figure 1 illustrates that despite substantial loss of parameters to sparsification, the entropically estimated HMMs were, on average, better predictors of notes. (Each test sequence was truncated to a random length and the HMMs were used to predict the first missing note.) They also were better at discriminating between test chorales and temporally reversed test chorales—challenging because Bach famously employed melodic reversal as a compositional device. With larger models, parameter-trimming became state-trimming: An average of 1.6 states were "pinched off" the 35-state models when all incoming transitions were deleted.

While the conventionally estimated HMMs were wholly uninterpretable, in the entropically estimated HMMs one can discern several basic musical structures (figure 1, right), including self-transitioning states that output only tonic (C-E-G) or dominant (G-B-D) triads, lower- or upper-register diatonic tones (C-D-E or F-G-A-B), and mordents (A-♯G-A). We also found chordal state sequences (F-A-C) and states that lead to the tonic (C) via the mediant (E) or the leading tone (B).

**Handwriting:** We used 2D Gaussian-output HMMs to analyze handwriting data. Training data, obtained from the UNIPEN web site [Reynolds, 1992], consisted of sequences of normalized pen-position coordinates taken at 5msec intervals from 10 different individuals writing the digits 0-9. The HMMs were estimated from identical data and initial conditions (random upper-diagonal transition matrices; random output parameters). The diagrams in Figure 2 depict transition graphs of two HMMs modeling the pen-strokes for the digit "5," mapped onto the data. Ellipses indicate each state's output probability iso-contours (receptive field); ×s and arcs indicate state dwell and transition probabilities, respectively, by their thicknesses. Entropic estimation induces an interpretable automaton that captures essential structure and timing of the pen-strokes. 50 of the 80 original transition parameters

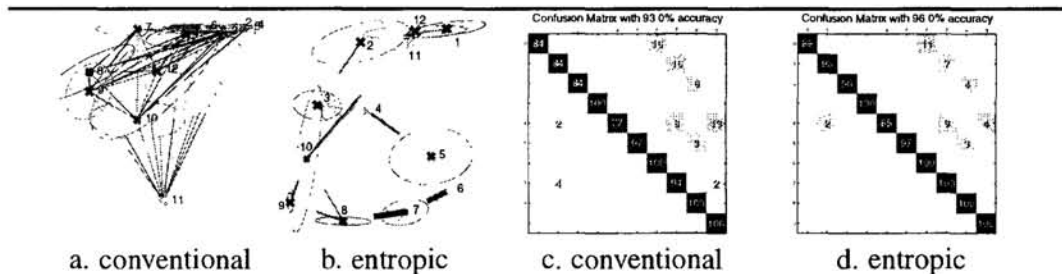

a. conventional     b. entropic     c. conventional     d. entropic

Figure 2: (a & b): State machines of conventionally and entropically estimated hidden Markov models of writing "5." (c & d): Confusion matrices for all digits.

were trimmed. Estimation without the entropic prior results in a wholly opaque model, in which none of the original dynamical parameters were trimmed. Model concision leads to better classification—the confusion matrices show cumulative classification error over ten trials with random initializations. Inspection of the parameters for the model in 2b showed that all writers began in states 1 or 2. From there it is possible to follow the state diagram to reconstruct the possible sequences of pen-strokes: Some writers start with the cap (state 1) while others start with the vertical (state 2); all loop through states 3-8 and some return to the top (via state 10) to add a horizontal (state 12) or diagonal (state 11) cap.

**Office activity:** Here we demonstrate a model of human activity learned from medium- to long-term ambient video. By activity, we mean spatio-temporal patterns in the pose, position, and movement of one's body. To make the vision tractable, we consider the activity of a single person in a relatively stable visual environment, namely, an office.

We track the gross shape and position of the office occupant by segmenting each image into foreground and background pixels. Foreground pixels are identified with reference to an acquired statistical model of the background texture and camera noise. Their ensemble properties such as motion or color are modeled via adaptive multivariate Gaussian distributions, re-estimated in each frame. A single bivariate Gaussian is fitted to the foreground pixels and we record the associated ellipse parameters [$mean_x$, $mean_y$, $\Delta mean_x$, $\Delta mean_y$, $mass$, $\Delta mass$, $elongation$, $eccentricity$]. Sequences of these observation vectors are used to train and test the HMMs.

Approximately 30 minutes of data were taken at 5Hz from an SGI IndyCam. Data was collected automatically and at random over several days by a program that started recording whenever someone entered the room after it had been empty 5+ minutes. Backgrounds were re-learned during absences to accommodate changes in lighting and room configuration. Prior to training, HMM states were initialized to tile the image with their receptive fields, and transition probabilities were initialized to prefer motion to adjoining tiles. Three sequences ranging from 1000 to 1900 frames in length were used for entropic training of 12, 16, 20, 25, and 30-state HMMs.

Entropic training yielded a substantially sparsified model with an easily interpreted state machine (see figure 3). Grouping of states into activities (done only to improve readability) was done by adaptive clustering on a proximity matrix which combined Mahalonobis distance and transition probability between states. The labels are the author's description of the set of frames claimed by each state cluster during forward-backward analysis of test data. Figure 4 illustrates this analysis, showing frames from a test sequence to which specific states are strongly tuned. State 5 (figure 3 right) is particularly interesting—it has a very non-specific receptive field, no self-transition, and an extremely low rate of occupancy. Instead of modeling data, it serves to *compress* the model by summarizing transition patterns that are common to several other states. The entropic model has proven to be quite superior for segmented new video into activities and detecting anomalous behavior.

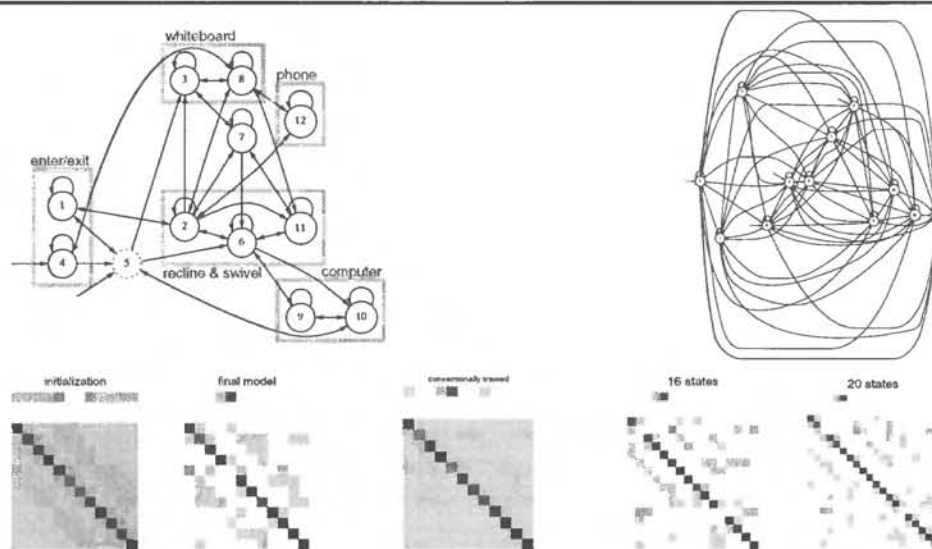

Figure 3: **Top:** The state machine found by entropic training (left) is easily labeled and interpreted. The state machine found by conventional training (right) is not, begin fully connected. **Bottom:** Transition matrices after (1) initialization, (2) entropic training, (3) conventional training, and (4 & 5) entropic training from larger initializations. The top row indicates initial probabilities of each state; each subsequent row indicates the transition probabilities out of a state. Color key: □ = 0; ■ = 1. The state machines above are extracted from 2 & 3. Note that 4 & 5 show the same qualitative structure as 2, but sparser, while 3 shows no almost no structure at all.

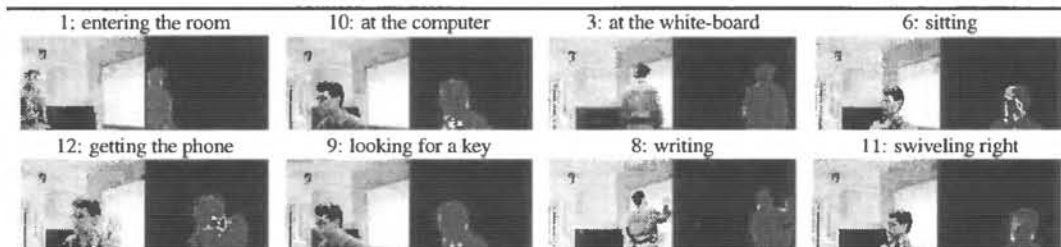

Figure 4: Some sample frames assigned high state-specific probabilities by the model. Note that some states are tuned to velocities, hence the difference between states 6 and 11.

## 4   Related work

**HMMs:** The literature of structure-learning in HMMs is based almost entirely on generate-and-test algorithms. These algorithms work by merging [Stolcke and Omohundro, 1994] or splitting [Takami and Sagayama, 1991] states, then retraining the model to see if any advantage has been gained. Space constraints force us to summarize a recent literature review: There are now more than 20 variations and improvements on these approaches, plus some heuristic constructive algorithms (e.g., [Wolfertstetter and Ruske, 1995]). Though these efforts use a variety of heuristic techniques and priors (including MDL) to avoid detrimental model changes, much of the computation is squandered and reported run-times often range from hours to days. Entropic estimation is exact, monotonic, and orders of magnitude faster—only slightly longer than standard EM parameter estimation.

**MDL:** Description length minimization is typically done via gradient ascent or search via model comparison; few estimators are known. Rissanen [1989] introduced an estimator for binary fractions, from which Vovk [1995] derived an approximate estimator for Bernoulli

models over discrete sample spaces. It approximates a special case of our exact estimator, which handles multinomial models in continuous sample spaces. Our framework provides a unified Bayesian framework for two issues that are often treated separately in MDL: estimating the number of parameters and estimating their values.

**MaxEnt:** Our prior has different premises and an effect opposite that of the "standard" MaxEnt prior $e^{-\alpha D(\theta \| \theta_0)}$. Nonetheless, our prior can be derived via MaxEnt reasoning from the premise that the expectation of the perplexity over all possible models is finite [Brand, 1998]. More colloquially, we almost always expect there to be learnable structure.

**Extensions:** For simplicity of exposition (and for results that are independent of model class), we have assumed prior independence of the parameters and taken $H(\theta)$ to be the combined parameter entropies of the model's component distributions. Depending on the model class, we can also provide variants of eqns. 1-8 for $H(\theta)$ =conditional entropy or $H(\theta)$ =entropy rate of the model. In Brand [1998] we present entropic MAP estimators for spread and covariance parameters with applications to mixtures-of-Gaussians, radial basis functions, and other popular models. In the same paper we generalize eqns. 1-8 with a temperature term, obtaining a MAP estimator that minimizes the free energy of the model. This folds deterministic annealing into EM, turning it into a quasi-global optimizer. It also provides a workaround for one known limitation of entropy minimization: It is inappropriate for learning from data that is atypical of the source process.

**Open questions:** Our framework is currently agnostic w.r.t. two important questions: Is there an optimal trimming policy? Is there a best entropy measure? Other questions naturally arise: Can we use the entropy to estimate the peakedness of the posterior distribution, and thereby judge the appropriateness of MAP models? Can we also directly minimize the entropy of the hidden variables, thereby obtaining discriminant training?

## 5 Conclusion

Entropic estimation is highly efficient hillclimbing procedure for simultaneously estimating model structure and parameters. It provides a clean Bayesian framework for minimizing all entropies associated with modeling, and an E-MAP algorithm that brings the structure of a randomly initialized model into alignment with hidden structures in the data via parameter extinction. The applications detailed here are three of many in which entropically estimated models have consistently outperformed maximum likelihood models in classification and prediction tasks. Most notably, it tends to produce interpretable models that shed light on the structure of relations between hidden variables and observed effects.

## References

Brand, M. (1997). Structure discovery in conditional probability models via an entropic prior and parameter extinction. *Neural Computation* To appear; accepted 8/98.

Brand, M. (1998). Pattern discovery via entropy minimization. To appear in *Proc., Artificial Intelligence and Statistics #7*.

Merz, C. and Murphy, P. (1998). UCI repository of machine learning databases.

Rabiner, L. R. (1989). A tutorial on hidden Markov models and selected applications in speech recognition. *Proceedings of the IEEE*, 77(2):257–286.

Reynolds, D. (1992). Handwritten digit data. UNIPEN web site, http://hwr.nici.kun.nl/unipen/. Donated by HP Labs, Bristol, England.

Rissanen, J. (1989). *Stochastic Complexity and Statistical Inquiry*. World Scientific.

Stolcke, A. and Omohundro, S. (1994). Best-first model merging for hidden Markov model induction. TR-94-003, International Computer Science Institute, U.C. Berkeley.

Takami, J.-I. and Sagayama, S. (1991). Automatic generation of the hidden Markov model by successive state splitting on the contextual domain and the temporal domain. TR SP91-88, IEICE.

Vovk, V. G. (1995). Minimum description length estimators under the optimal coding scheme. In Vitányi, P., editor, *Proc. Computational Learning Theory / Europe*, pages 237–251. Springer-Verlag.

Wolfertstetter, F. and Ruske, G. (1995). Structured Markov models for speech recognition. In *International Conference on Acoustics, Speech, and Signal Processing*, volume 1, pages 544–7.
